# Reverse Multi-Label Learning

**James Petterson**
NICTA, Australian National University
Canberra, ACT, Australia
james.petterson@nicta.com.au

**Tiberio Caetano**
NICTA, Australian National University
Canberra, ACT, Australia
tiberio.caetano@nicta.com.au

## Abstract

Multi-label classification is the task of predicting potentially multiple labels for a given instance. This is common in several applications such as image annotation, document classification and gene function prediction. In this paper we present a formulation for this problem based on *reverse* prediction: we predict sets of instances given the labels. By viewing the problem from this perspective, the most popular quality measures for assessing the performance of multi-label classification admit relaxations that can be efficiently optimised. We optimise these relaxations with standard algorithms and compare our results with several state-of-the-art methods, showing excellent performance.

## 1 Introduction

Recently, multi-label classification (MLC) has been drawing increasing attention from the machine learning community (e.g., [1, 2, 3, 4]). Unlike in the case of multi-class learning, in MLC each instance may belong to multiple classes simultaneously. This reflects the situation in many real-world problems: in document classification, one document can cover multiple subjects; in biology, a gene can be associated with a set of functional classes [5]; in image annotation, one image can have several tags [6].

As diverse as the applications, however, are the evaluation measures used to assess the performance of different methods. That is understandable, since different applications have different goals. In e-discovery applications [7] it is mandatory that all relevant documents are retrieved, so recall is the most relevant measure. In web search, on the other hand, precision is also important, so the $F_1$-score, which is the harmonic mean of precision and recall, might be more appropriate.

In this paper we present a method for MLC which is able to optimise appropriate surrogates for a variety of performance measures. This means that the objective function being optimised by the method is tailored to the performance measure on which we want to do well in our specific application. This is in contrast particularly with probabilistic approaches, which typically aim for maximisation of likelihood scores rather than the performance measure used to assess the quality of the results. In addition, the method is based on well-understood facts from the domain of structured output learning, which gives us theoretical guarantees regarding the accuracy of the results obtained. Finally, source code is made available by us.

An interesting aspect of the method is that we are only able to optimise the desired performance measures because we formulate the prediction problem in a *reverse* manner, in the spirit of [8]. We pose the prediction problem as predicting sets of instances given the labels. When this insight is fit into max-margin structured output methods, we obtain surrogate losses for the most widely used performance measures for multi-label classification. We perform experiments against state-of-the-art methods in five publicly available benchmark datasets for MLC, and the proposed approach is the best performing overall.

### 1.1 Related Work

The literature in this topic is vast and we cannot possibly make justice here since a comprehensive review is clearly impractical. Instead, we focus particularly on some state-of-the-art approaches

that have been tested on publicly available benchmark datasets for MLC, which facilitates a fair comparison against our method. A straightforward way to deal with multiple labels is to solve a binary classification problem for each one of them, treating them independently. This approach is known as *Binary Method* (BM) [9]. *Classifier Chains* (CC) [4] extends that by building a chain of binary classifiers, one for each possible label, but with each classifier augmented by all prior relevance predictions. Since the order of the classifiers in the chain is arbitrary, the authors also propose an ensemble method – *Ensemble of Classifier Chains* (ECC) – where several random chains are combined with a voting scheme. *Probabilistic Classifier Chains* (PCC) [1] extends CC to the probabilistic setting, with EPCC [1] being its corresponding ensemble method. Another way of working with multiple labels is to consider each possible set of labels as a class, thus encoding the problem as single-label classification. The problem with that is the exponentially large number of classes. *RAndom K-labELsets* (RAKEL) [10] deals with that by proposing an ensemble of classifiers, each one taking a small random subset of the labels and learning a single-label classifier for the prediction of each element in the power set of this subset. Other proposed ensemble methods are *Ensemble of Binary Method* (EBM) [4], which applies a simple voting scheme to a set of BM classifiers, and *Ensemble of Pruned Sets* (EPS) [11], which combines a set of Pruned Sets (PS) classifiers. PS is essentially a problem transformation method that maps sets of labels to single labels while pruning away infrequently occurring sets.*Canonical Correlation Analysis* (CCA) [3] exploits label relatedness by using a probabilistic interpretation of CCA as a dimensionality reduction technique and applying it to learn useful predictive features for multi-label learning. *Meta Stacking* (MS) [12] also exploits label relatedness by combining text features and features indicating relationships between classes in a discriminative framework.

Two papers closely related to ours from the methodological point of view, which are however not tailored particularly to the multi-label learning problem, are [13] and [14]. In [13] the author proposes a smooth but non-concave relaxation of the $F$-measure for binary classification problems using a logistic regression classifier, and optimisation is performed by taking the maximum across several runs of BFGS starting from random initial values. In [14] the author proposes a method for optimising multivariate performance measures in a general setting in which the loss function is not assumed to be additive in the instances nor in the labels. The method also consists of optimising a convex relaxation of the derived losses. The key difference of our method is that we have a specialised convex relaxation for the case in which the loss does not decompose over the instances, but *does* decompose over the labels.

## 2   The Model

Let the input $x \in \mathcal{X}$ denote a label (e.g., a tag of an image), and the output $y \in \mathcal{Y}$ denote a *set of instances*, (e.g., a set of training images). Let $N = |\mathcal{X}|$ be the number of labels and $V$ be the number of instances. An input label $x$ is encoded as $x \in \{0,1\}^N$, s.t. $\sum_i x_i = 1$. For example if $N = 5$ the second label is denoted as $x = [0\ 1\ 0\ 0\ 0]$. An output instance $y$ is encoded as $y \in \{0,1\}^V (\mathcal{Y} := \{0,1\}^V)$, and $y_i^n = 1$ iff instance $x^n$ was annotated with label $i$. For example if $V = 10$ and only instances 1 and 3 are annotated with label 2, then the $y$ corresponding to $x = [0\ 1\ 0\ 0\ 0]$ is $y = [1\ 0\ 1\ 0\ 0\ 0\ 0\ 0\ 0\ 0]$. We assume a given training set $\{(x^n, y^n)\}_{n=1}^N$, where $\{x^n\}_{n=1}^N$ comprises the entirety of labels available ($\{x^n\}_{n=1}^N = \mathcal{X}$), and $\{y^n\}_{n=1}^N$ represents the sets of instances associated to those labels. The task consists of estimating a map $f : \mathcal{X} \to \mathcal{Y}$ which reproduces well the outputs of the training set (i.e., $f(x^n) \approx y^n$) but also generalises well to new test instances.

### 2.1   Loss Functions

The reason for this *reverse prediction* is the following: most widely accepted performance measures target information retrieval (IR) applications – that is, given a label we want to find a set of relevant instances. As a consequence, the measures are *averaged over the set of possible labels*. This is the case for, in particular, *Macro-precision*, *Macro-recall*, *Macro-$F_\beta$*[1] and *Hamming loss* [10]:

$$\text{Macro-precision} = \frac{1}{N} \sum_{n=1}^N p(y^n, \bar{y}^n), \qquad \text{Macro-recall} = \frac{1}{N} \sum_{n=1}^N r(y^n, \bar{y}^n)$$

$$\text{Macro-F}_\beta = \frac{1}{N} \sum_{n=1}^{N} (1+\beta^2) \frac{p(y^n, \bar{y}^n) r(y^n, \bar{y}^n)}{\beta^2 p(y^n, \bar{y}^n) + r(y^n, \bar{y}^n)}, \quad \text{Hamming loss} = \frac{1}{N} \sum_{n=1}^{N} h(y^n, \bar{y}^n),$$

where

$$h(y, \bar{y}) = \frac{y^T \mathbf{1} + \bar{y}^T \mathbf{1} - 2y^T \bar{y}}{V}, \quad p(y, \bar{y}) = \frac{y^T \bar{y}}{\bar{y}^T \bar{y}}, \quad r(y, \bar{y}) = \frac{y^T \bar{y}}{y^T y}.$$

Here, $\bar{y}^n$ is our prediction for input label $n$, and $y^n$ the corresponding ground-truth. Since these measures average over the labels, in order to optimise them we need to average over the labels as well, and this happens naturally in a setting in which the empirical risk is additive on the labels.[2]

Instead of maximising a performance measure we frame the problem as minimising a loss function associated to the performance measure. We assume a known loss function $\Delta : \mathcal{Y} \times \mathcal{Y} \to \mathbb{R}_+$ which assigns a non-negative number to every possible pair of outputs. This loss function represents how much we want to penalise a prediction $\bar{y}$ when the correct prediction is $y$, i.e., it has the opposite semantics of a performance measure. As already mentioned, we will be able to deal with a variety of loss functions in this framework, but for concreteness of exposition we will focus on a loss derived from the *Macro-F$_\beta$* score defined above, whose particular case for $\beta$ equal to 1 ($F_1$) is arguably the most popular performance measure for multi-label classification. In our notation, the $F_\beta$ score of a given prediction is

$$F_\beta(y, \bar{y}) = (1+\beta^2) \frac{y^T \bar{y}}{\beta^2 y^T y + \bar{y}^T \bar{y}}, \tag{1}$$

and since $F_\beta$ is a score of *alignment* between $y$ and $\bar{y}$, one possible choice for the loss is $\Delta(y, \bar{y}) = 1 - F_\beta(y, \bar{y})$, which is the one we focus on in this paper,

$$\Delta(y, \bar{y}) = 1 - (1+\beta^2) \frac{y^T \bar{y}}{\beta^2 y^T y + \bar{y}^T \bar{y}}. \tag{2}$$

## 2.2 Features and Parameterization

Our next assumption is that the prediction for a given input $x$ returns the maximiser(s) of a linear score of the model parameter vector $\theta$, i.e., a prediction is given by $\bar{y}$ such that [3]

$$\bar{y} \in \operatorname*{argmax}_{y \in \mathcal{Y}} \langle \phi(x, y), \theta \rangle. \tag{3}$$

Here we assume that $\phi(x, y)$ is linearly composed of features of the instances encoded in each $y_v$, i.e., $\phi(x, y) = \sum_{v=1}^{V} y_v (\psi_v \otimes x)$. The vector $\psi_v$ is the feature representation for the instance $v$. The map $\phi(x, y)$ will be the zero vector whenever $y_v = 0$, i.e., when instance $v$ does not have label $x$. The feature map $\phi(x, y)$ has a total of $DN$ dimensions, where $D$ is the dimensionality of our instance features ($\psi_v$) and $N$ is the number of labels. Therefore $DN$ is the dimensionality of our parameter $\theta$ to be learned.

## 2.3 Optimisation Problem

We are now ready to formulate our estimator. We assume an initial, 'ideal' estimator taking the form

$$\theta^* = \operatorname*{argmin}_{\theta} \left[ \left( \frac{1}{N} \sum_{n=1}^{N} \Delta(\bar{y}^n(x^n; \theta), y^n) \right) + \frac{\lambda}{2} \|\theta\|^2 \right]. \tag{4}$$

In other words, we want to find a model that minimises the average prediction loss in the training set *plus* a quadratic regulariser that penalises complex solutions (the parameter $\lambda$ determines the trade-off between data fitting and good generalisation). Estimators of this type are known as regularised risk minimisers [15].

# 3 Optimisation

## 3.1 Convex Relaxation

The optimisation problem (4) is non-convex. Even more critical, the loss is a piecewise constant function of $\theta$.[4] A similar problem occurs when one aims at optimising a $0/1$ loss in binary classification; in that case, a typical workaround consists of minimising a surrogate convex loss function which upper bounds the $0/1$ loss, for example the hinge loss, what gives rise to the support vector machine. Here we use an analogous approach, notably popularised in [16], which optimises a convex upper bound on the structured loss of (4). The resulting optimisation problem is

$$[\theta^*, \xi^*] = \operatorname*{argmin}_{\theta, \xi} \left[ \frac{1}{N} \sum_{n=1}^{N} \xi_n + \frac{\lambda}{2} \|\theta\|^2 \right] \tag{5}$$

$$\text{s.t. } \langle \phi(x^n, y^n), \theta \rangle - \langle \phi(x^n, y), \theta \rangle \geq \Delta(y, y^n) - \xi_n, \quad \xi_n \geq 0 \tag{6}$$

$$\forall n, y \in \mathcal{Y}.$$

It is easy to see that $\xi_n^*$ upper bounds $\Delta(\bar{y}_*^n, y^n)$ (and therefore the objective in (5) upper bounds that of (4) for the optimal solution). Here, $\bar{y}_*^n := \operatorname{argmax}_y \langle \phi(x^n, y), \theta^* \rangle$. First note that since the constraints (6) hold for all $y$, they also hold for $\bar{y}_*^n$. Second, the left hand side of the inequality for $y = \bar{y}^n$ must be non-positive from the definition of $\bar{y}$ in equation (3). It then follows that $\xi_n^* \geq \Delta(\bar{y}_*^n, y^n)$.

The constraints (6) basically enforce a loss-sensitive margin: $\theta$ is learned so that mispredictions $y$ that incur some loss end up with a score $\langle \phi(x^n, y), \theta \rangle$ that is smaller than the score $\langle \phi(x^n, y^n), \theta \rangle$ of the correct prediction $y^n$ by a margin equal to that loss (minus slack $\xi$). The formulation is a generalisation of support vector machines for the case in which there are an exponential number of classes $y$. It is in this sense that our approach is somewhat related in spirit to [10], as mentioned in the Introduction. However, as described below, here we can use a method for selecting a polynomial number of constraints which provably approximates well the original problem.

The optimisation problem (5) has $n|\mathcal{Y}| = n2^V$ constraints. Naturally, this number is too large to allow for a practical solution of the quadratic program. Here we resort to a constraint generation strategy, which consists of starting with no constraints and iteratively adding the most violated constraint for the current solution of the optimisation problem. Such an approach is assured to find an $\epsilon$-close approximation of the solution of (5) after including only $O(\epsilon^{-2})$ constraints [16]. The key problem that needs to be solved at each iteration is *constraint generation*, i.e., to find the maximiser of the violation margin $\xi_n$,

$$y_n^* \in \operatorname*{argmax}_{y \in \mathcal{Y}} \left[ \Delta(y, y^n) + \langle \phi(x^n, y), \theta \rangle \right]. \tag{7}$$

The difficulty in solving the above optimisation problem depends on the choice of $\phi(x, y)$ and $\Delta$. Next we investigate how this problem can be solved for our particular choices of these quantities.

## 3.2 Constraint generation

Using eq.(2) and $\phi(x, y) = \sum_{v=1}^{V} y_v(\psi_v \otimes x)$, eq. (7) becomes

$$y_n^* \in \operatorname*{argmax}_{y \in \mathcal{Y}} \langle y, z_n \rangle. \tag{8}$$

where

$$z_n = \Psi\theta^n - \frac{(1 + \beta^2)y^n}{\|y\|^2 + \beta^2 \|y^n\|^2}, \tag{9}$$

and

- $\Psi$ is a $V \times D$ matrix with row $v$ corresponding to $\psi_v$;
- $\theta^n$ is the $n^{th}$ column of matrix $\theta$;

**Algorithm 1** Reverse Multi-Label Learning

---

1: **Input:** training set $\{(x^n, y^n)\}_{n=1}^N$, $\lambda$, $\beta$, **Output:** $\theta$
2: Initialize $i = 1$, $\theta_1 = 0$, MAX$= -\infty$
3: **repeat**
4:     **for** $n = 1$ **to** $N$ **do**
5:         Compute $y_n^*$ (Naïve: Algorithm 2. Improved: See Appendix)
6:     **end for**
7:     Compute gradient $g_i$ (equation (12)) and objective $o_i$ (equation (11))
8:     $\theta_{i+1} := \operatorname{argmin}_\theta \frac{\lambda}{2} \|\theta\|^2 + \max(0, \max_{j \leq i} \langle g_j, \theta \rangle + o_j)$; $i \leftarrow i + 1$
9: **until** converged (see [18])
10: **return** $\theta$

---

---

**Algorithm 2** Naïve Constraint Generation

---

1: **Input:** $(x^n, y^n)$, $\Psi$, $\theta$, $\beta$, $V$, **Output:** $y_n^*$
2: MAX$= -\infty$
3: **for** $k = 1$ **to** $V$ **do**
4:     $z_n = \Psi \theta^n - \frac{(1+\beta^2)y^n}{k + \beta^2 \|y^n\|^2}$
5:     $y^* = \operatorname{argmax}_{y \in \mathcal{Y}_k} \langle y, z_n \rangle$ (i.e. find top $k$ entries in $z_n$ in $O(V)$ time)
6:     CURRENT$= \max_{y \in \mathcal{Y}_k} \langle y, z_n \rangle$
7:     **if** CURRENT>MAX **then**
8:         MAX = CURRENT
9:         $y_n^* = y^*$
10:     **end if**
11: **end for**
12: **return** $y_n^*$

---

We now investigate how to solve (8) for a fixed $\theta$. For the purpose of clarity, here we describe a simple, naïve algorithm. In the appendix we present a more involved but much faster algorithm. A simple algorithm can be obtained by first noticing that $z_n$ depends on $y$ only through the number of its nonzero elements. Consider the set of all $y$ with precisely $k$ nonzero elements, i.e., $\mathcal{Y}_k =: \{y : \|y\|^2 = k\}$. Then the objective in (8), if the maximisation is instead restricted to the domain $\mathcal{Y}_k$, is effectively *linear* in $y$, since $z_n$ in this case is a constant w.r.t. $y$. Therefore we can solve separately for each $\mathcal{Y}_k$ by finding the top $k$ entries in $z_n$. Finding the top $k$ elements of a list of size $V$ can be done in $O(V)$ time [17]. Therefore we have a $O(V^2)$ algorithm (for every $k$ from 1 to $V$, solve $\operatorname{argmax}_{y \in \mathcal{Y}_k} \langle y, z \rangle$ in $O(V)$ expected time). Algorithm 1 describes in detail the optimisation, as solved by BMRM [18], and Algorithm 2 shows the naïve constraint generation routine. The BMRM solver requires both the value of the objective function for the slack corresponding to the most violated constraint and its gradient. The value of the slack variable corresponding to $y_n^*$ is

$$\xi_n^* = \Delta(y_n^*, y^n) + \langle \phi(x^n, y_n^*), \theta \rangle - \langle \phi(x^n, y^n), \theta \rangle, \tag{10}$$

thus the objective function from (5) becomes

$$\frac{1}{N} \sum_n \Delta(y_n^*, y^n) + \langle \phi(x^n, y_n^*), \theta \rangle - \langle \phi(x^n, y^n), \theta \rangle + \frac{\lambda}{2} \|\theta\|^2, \tag{11}$$

whose gradient (with respect to $\theta$) is

$$\lambda \theta - \frac{1}{N} \sum_n (\phi(x^n, y^n) - \phi(x^n, y_n^*)). \tag{12}$$

We need both expressions (11) and (12) in Algorithm 1.

### 3.3 Prediction at Test Time

The problem to be solved at test time (eq. (3)) has the same form as the problem of constraint generation (eq. (7)), the only difference being that $z_n = \Psi \theta^n$ (i.e., the second term in eq. (9), due to the loss, is not present). Since $z_n$ a constant vector, the solution $y_n^*$ for (7) is the vector that indicates the positive entries of $z_n$, which can be efficiently found in $O(V)$. Therefore inference at prediction time is very fast.

Table 1: Evaluation scores and corresponding losses

| score | $\Delta(y, \bar{y})$ |
|---|---|
| macro-$F_\beta$ | $1 - \frac{(1+\beta^2)(y^T \bar{y})}{\beta^2 y^T y + \bar{y}^T \bar{y}}$ |
| macro-precision | $1 - \frac{y^T \bar{y}}{\bar{y}^T \bar{y}}$ |
| macro-recall | $1 - \frac{y^T \bar{y}}{y^T y}$ |
| Hamming loss | $\frac{y^T \mathbf{1} + \bar{y}^T \mathbf{1} - 2 y^T \bar{y}}{V}$ |

Table 2: Datasets. #train/#test denotes the number of observations used for training and testing respectively; $N$ is the number of labels and $D$ the dimensionality of the features.

| dataset | domain | #train | #test | N | D |
|---|---|---|---|---|---|
| yeast | biology | 1500 | 917 | 14 | 103 |
| scene | image | 1211 | 1196 | 6 | 294 |
| medical | text | 645 | 333 | 45 | 1449 |
| enron | text | 1123 | 579 | 53 | 1001 |
| emotions | music | 391 | 202 | 6 | 72 |

### 3.4 Other scores

Up to now we have focused on optimising Macro-$F_\beta$, which already gives us several scores, in particular Macro-$F_1$, macro-recall and macro-precision. We can however optimise other scores, in particular the popular Hamming loss – Table 1 shows a list with the corresponding loss, which we then plug in eq.(4).

Note that for *Hamming loss* and *macro-recall* the denominator is constant, and therefore it is not necessary to solve (8) multiple times as described earlier, which makes constraint generation as fast as test-time prediction (see subsection 3.3).

## 4 Experimental Results

In this section we evaluate our method in several real world datasets, for both *macro-$F_\beta$* and *Hamming loss*. These scores were chosen because macro-$F_\beta$ is a generalisation of the most relevant scores, and the Hamming loss is a generic, popular score in the multi-label classification literature.

### Datasets

We used 5 publicly available[5] multi-label datasets: *yeast*, *scene*, *medical*, *enron* and *emotions*. We selected these datasets because they cover a variety of application domains – biology, image, text and music – and there are published results of competing methods on them for some of the popular evaluation measures for MLC (*macro-$F_1$* and *Hamming loss*). Table 2 describes them in more detail.

### Model selection

Our model requires only one parameter: $\lambda$, the trade-off between data fitting and good generalisation. For each experiment we selected it with 5-fold cross-validation using only the training data.

### Implementation

Our implementation is in C++, using the *Bundle Methods for Risk Minimization* (BMRM) of [18] as a base. Source code is available[6] under the Mozilla Public License.[7]

### Comparison to published results on Macro-$F_1$

In our first set of experiments we compared our model to published results on the Macro-$F_1$ score. We strived to make our comparison as broad as possible, but we limited ourselves to methods with published results on public datasets, where the experimental setting was described in enough detail to allow us to make a fair comparison.

We therefore compared our model to Canonical Correlation Analysis [3] (CCA), Binary Method [9] (BM), Classifier Chains [4] (CC), Subset Mapping [19] (SM), Meta Stacking [12] (MS), Ensembles of Binary Method [4] (EBM) , Ensembles of Classifier Chains [4] (ECC), Ensembles of Pruned Sets [11] (EPS) and Random K Label Subsets [10] (RAKEL).

Table 3 summarizes our results, along with competing methods' which were taken from compilations by [3] and [4]. We can see that our model has the best performance in *yeast*, *medical* and *enron*. In

*scene* it doesn't perform as well – we suspect this is related to the label cardinality of this dataset: almost all instances have just one label, making this essentially equivalent to a multiclass dataset.

**Comparison to published results on Hamming Loss**

To illustrate the flexibility of our model we also evaluated it on the Hamming loss. Here, we compared our model to classifier chains [4] (CC), probabilistic classifier chains [1] (PCC), ensembles of classifier chains [4] (ECC) and ensembled probabilistic classifier chains [1] (EPCC). These are the methods for which we could find Hamming loss results on publicly available data.

Table 4 summarizes our results, along with competing methods' which were taken from a compilation by [1]. As can be seen, our model has the best performance on both datasets.

**Results on $F_\beta$**

One strength of our method is that it can be optimised for the specific measure we are interested in. In Macro-$F_\beta$, for example, $\beta$ is a trade-off between *precision* and *recall*: when $\beta \to 0$ we recover *precision*, and when $\beta \to \infty$ we get *recall*. Unlike with other methods, given a desired precision/recall trade-off encoded in a choice of $\beta$, we can optimise our model such that it gets the best performance on Macro-$F_\beta$. To show this we ran our method on all five datasets, but this time with different choices of $\beta$, ranging from $10^{-2}$ to $10^2$. In this case, however, we could not find published results to compare to, so we used Mulan[8], an open-source library for learning from multi-label datasets, to train three models: BM[9], RAKEL[10] and MLKNN[20]. BM was chosen as a simple baseline, and RAKEL and MLKNN are the two state-of-the-art methods available in the package.

MLKNN has two parameters: the number of neighbors $k$ and a smoothing parameter $s$ controlling the strength of the uniform prior. We kept both fixed to 10 and 1.0, respectively, as was done in [20]. RAKEL has three parameters: the number of models $m$, the size of the labelset $k$ and the threshold $t$. Since a complete search over the parameter space would be impractical, we adopted the library's default for $t$ and $m$ (respectively 0.5 and $2 * N$) and set $k$ to $\frac{N}{2}$ as suggested by [4]. For BM we kept the library's defaults.

In Figure 1 we plot the results. We can see that BM tends to prioritize *recall* (right side of the plot), while ML-KNN and RAKEL give more emphasis to *precision* (left side). Our method, however, goes well in both sides, as it is trained separately for each value of $\beta$. In both *scene* and *yeast* it dominates the right side while is still competitive on the left side. And in the other three datasets – *medical*, *enron* and *emotions* – it practically dominates over the entire range of $\beta$.

## 5   Conclusion and Future Work

We presented a new approach to multi-label learning which consists of predicting sets of instances from the labels. This apparent unintuitive approach is in fact natural since, once the problem is viewed from this perspective, many popular performance measures admit convex relaxations that can be directly and efficiently optimised with existing methods. The method only requires one parameter, as opposed to most existing methods, which have several. The method leverages on existing tools from structured output learning, which gives us certain theoretical guarantees. A simple version of constraint generation is presented for small problems, but we also developed a scalable, fast version for dealing with large datasets. We presented a detailed experimental comparison against several state-of-the-art methods and overall our performance is notably superior.

A fundamental limitation of our current approach is that it does not handle dependencies among labels. It is however possible to include such dependencies by assuming for example a bivariate feature map on the labels, rather than univariate. This however complicates the algorithmics, and is left as subject for future research.

## Acknowledgements

We thank Miro Dudík as well as the anonymous reviewers for insightful observations that helped to improve the paper. NICTA is funded by the Australian Government as represented by the Department of Broadband, Communications and the Digital Economy and the Australian Research Council through the ICT Centre of Excellence program.

Table 3: Macro-F$_1$ results. Bold face indicates the best performance. We don't have results for CCA in the Medical and Enron datasets.

| Dataset | Ours | CCA | CC | BM | SM | MS | ECC | EBM | EPS | RAKEL |
|---------|------|-----|-----|-----|-----|-----|-----|-----|-----|-------|
| Yeast | **0.440** | 0.346 | 0.346 | 0.326 | 0.327 | 0.331 | 0.362 | 0.364 | 0.420 | 0.413 |
| Scene | 0.671 | 0.374 | 0.696 | 0.685 | 0.666 | 0.694 | 0.742 | 0.729 | **0.763** | 0.750 |
| Medical | **0.420** | - | 0.377 | 0.364 | 0.321 | 0.370 | 0.386 | 0.382 | 0.324 | 0.377 |
| Enron | **0.243** | - | 0.198 | 0.197 | 0.144 | 0.198 | 0.201 | 0.201 | 0.155 | 0.206 |

Table 4: Hamming loss results. Bold face indicates the best performance.

| Dataset | Ours | CC | PCC | ECC | EPCC |
|---------|------|-----|-----|-----|------|
| Scene | **0.1271** | 0.1780 | 0.1780 | 0.1503 | 0.1498 |
| Emotions | **0.2252** | 0.2448 | 0.2417 | 0.2428 | 0.2372 |

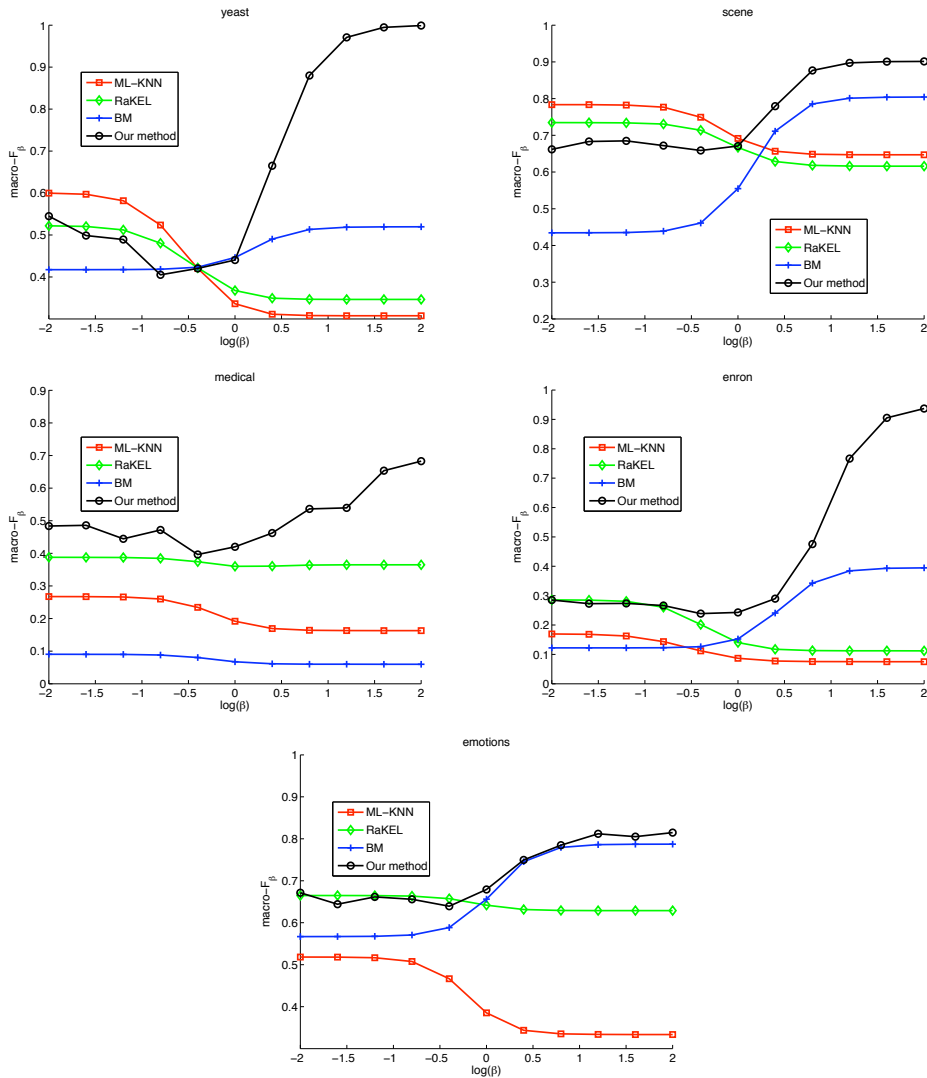

Figure 1: Macro-F$_\beta$ results on five datasets, with $\beta$ ranging from $10^{-2}$ to $10^2$ (i.e., $\log_{10} \beta$ ranging from -2 to 2). The center point ($\log \beta = 0$) corresponds to macro-F$_1$. $\beta$ controls a trade-off between *Macro-precision* (left side) and *Macro-recall* (right side).

## Footnotes

[1]Macro-$F_1$ is the particular case of this when $\beta$ equals to 1. Macro-precision and macro-recall are particular cases of macro-$F_\beta$ for $\beta \to 0$ and $\beta \to \infty$, respectively.

[2]The Hamming loss also averages over the instances so it can be optimised in the 'normal' (not reverse) direction as well.

[3]$\langle A, B \rangle$ denotes the inner product of the vectorized versions of $A$ and $B$

[4]There is a countable number of loss values but an uncountable number of parameters, so there are large equivalence classes of parameters that correspond to precisely the same loss.

[5]http://mulan.sourceforge.net/datasets.html

[6]http://users.cecs.anu.edu.au/~jpetterson/.

[7]http://www.mozilla.org/MPL/MPL-1.1.html

[8]http://mulan.sourceforge.net/

# References

[1] Krzysztof Dembczynski, Weiwei Cheng, and Eyke Hüllermeier. Bayes Optimal Multilabel Classification via Probabilistic Classifier Chains. In *Proc. Intl. Conf. Machine Learning*, 2010.

[2] Xinhua Zhang, T. Graepel, and Ralf Herbrich. Bayesian Online Learning for Multi-label and Multi-variate Performance Measures. In *Proc. Intl. Conf. on Artificial Intelligence and Statistics*, volume 9, pages 956–963, 2010.

[3] Piyush Rai and Hal Daume. Multi-Label Prediction via Sparse Infinite CCA. In Y. Bengio, D. Schuurmans, J. Lafferty, C. K. I. Williams, and A. Culotta, editors, *Advances in Neural Information Processing Systems 22*, pages 1518–1526. 2009.

[4] Jesse Read, Bernhard Pfahringer, Geoffrey Holmes, and Eibe Frank. Classifier chains for multi-label classification. In Wray L. Buntine, Marko Grobelnik, Dunja Mladenic, and John Shawe-Taylor, editors, *ECML/PKDD (2)*, volume 5782 of *Lecture Notes in Computer Science*, pages 254–269. Springer, 2009.

[5] André Elisseeff and Jason Weston. A kernel method for multi-labelled classification. In *Annual ACM Conference on Research and Development in Information Retrieval*, pages 274–281, 2005.

[6] Matthieu Guillaumin, Thomas Mensink, Jakob Verbeek, and Cordelia Schmid. TagProp: Discriminative Metric Learning in Nearest Neighbor Models for Image Auto-Annotation. In *Proc. Intl. Conf. Computer Vision*, 2009.

[7] Douglas W. Oard and Jason R. Baron. Overview of the TREC 2008 Legal Track.

[8] Linli Xu, Martha White, and Dale Schuurmans. Optimal reverse prediction. *Proc. Intl. Conf. Machine Learning*, pages 1–8, 2009.

[9] Grigorios Tsoumakas, Ioannis Katakis, and Ioannis P. Vlahavas. *Mining Multi-label Data*. Springer, 2009.

[10] Grigorios Tsoumakas and Ioannis P. Vlahavas. Random k-labelsets: An ensemble method for multilabel classification. In *Proceedings of the 18th European Conference on Machine Learning (ECML 2007)*, pages 406–417, Warsaw, Poland, 2007.

[11] Jesse Read, Bernhard Pfahringer, and Geoff Holmes. Multi-label classification using ensembles of pruned sets. In *ICDM '08: Proceedings of the 2008 Eighth IEEE International Conference on Data Mining*, pages 995–1000, Washington, DC, USA, 2008. IEEE Computer Society.

[12] Shantanu Godbole and Sunita Sarawagi. Discriminative methods for multi-labeled classification. In *Proceedings of the 8th Pacific-Asia Conference on Knowledge Discovery and Data Mining*, pages 22–30. Springer, 2004.

[13] Martin Jansche. Maximum expected F-measure training of logistic regression models. *HLT*, pages 692–699, 2005.

[14] T. Joachims. A support vector method for multivariate performance measures. In *Proc. Intl. Conf. Machine Learning*, pages 377–384, San Francisco, California, 2005. Morgan Kaufmann Publishers.

[15] V. Vapnik. *Statistical Learning Theory*. John Wiley and Sons, New York, 1998.

[16] I. Tsochantaridis, T. Joachims, T. Hofmann, and Y. Altun. Large margin methods for structured and interdependent output variables. *J. Mach. Learn. Res.*, 6:1453–1484, 2005.

[17] D. E. Knuth. *The Art of Computer Programming: Fundamental Algorithms*, volume 1. Addison-Wesley, Reading, Massachusetts, second edition, 1998.

[18] Choon Hui Teo, S. V. N. Vishwanathan, Alex J. Smola, and Quoc V. Le. Bundle methods for regularized risk minimization. *Journal of Machine Learning Research*, 11:311–365, 2010.

[19] Robert E. Schapire and Y. Singer. Improved boosting algorithms using confidence-rated predictions. *Machine Learning*, 37(3):297–336, 1999.

[20] Min-Ling Zhang and Zhi-Hua Zhou. ML-KNN: A lazy learning approach to multi-label learning. *Pattern Recognition*, 40(7):2038–2048, July 2007.

